# Accelerated Training for Matrix-norm Regularization: A Boosting Approach

**Xinhua Zhang**[*]**, Yaoliang Yu and Dale Schuurmans**
Department of Computing Science, University of Alberta, Edmonton AB T6G 2E8, Canada
{xinhua2,yaoliang,dale}@cs.ualberta.ca

## Abstract

Sparse learning models typically combine a smooth loss with a nonsmooth penalty, such as trace norm. Although recent developments in sparse approximation have offered promising solution methods, current approaches either apply only to matrix-norm *constrained* problems or provide suboptimal convergence rates. In this paper, we propose a boosting method for *regularized* learning that guarantees $\epsilon$ accuracy within $O(1/\epsilon)$ iterations. Performance is further accelerated by interlacing boosting with fixed-rank local optimization—exploiting a simpler local objective than previous work. The proposed method yields state-of-the-art performance on large-scale problems. We also demonstrate an application to latent multiview learning for which we provide the first efficient weak-oracle.

## 1 Introduction

Our focus in this paper is on unsupervised learning problems such as matrix factorization or latent subspace identification. Automatically uncovering latent factors that reveal important structure in data is a longstanding goal of machine learning research. Such an analysis not only provides understanding, it can also facilitate subsequent data storage, retrieval and processing. We focus in particular on coding or dictionary learning problems, where one seeks to decompose a data matrix $X$ into an approximate factorization $\hat{X} = UV$ that minimizes reconstruction error while satisfying other properties like low rank or sparsity in the factors. Since imposing a bound on the rank or number of non-zero elements generally makes the problem intractable, such constraints are usually replaced by carefully designed regularizers that promote low rank or sparse solutions [1–3].

Interestingly, for a variety of dictionary constraints and regularizers, the problem is equivalent to a matrix-norm regularized problem on the reconstruction matrix $\hat{X}$ [1, 4]. One intensively studied example is the trace norm, which corresponds to bounding the Euclidean norm of the code vectors in $U$ while penalizing $V$ via its $\ell_{21}$ norm. To solve trace norm regularized problems, variational methods that optimize over $U$ and $V$ only guarantee local optimality, while proximal gradient algorithms that operate on $\hat{X}$ [5, 6] can achieve an $\epsilon$ accurate (global) solutions in $O(1/\sqrt{\epsilon})$ iterations, but these require singular value thresholding [7] at each iteration, preventing application to large problems.

Recently, remarkable promise has been demonstrated for sparse approximation methods. [8] converts the trace norm problem into an optimization over positive semidefinite (PSD) matrices, then solves the problem via greedy sparse approximation [9, 10]. [11] further generalizes the algorithm from trace norm to gauge functions [12], dispensing with the PSD conversion. However, these schemes turn the regularization into a constraint. Despite their theoretical equivalence, many practical applications require the solution to the regularized problem, *e.g.* when nested in another problem.

In this paper, we optimize the regularized objective directly by reformulating the problem in the framework of $\ell_1$ penalized boosting [13, 14], allowing it to be solved with a general procedure developed in Section 2. Each iteration of this procedure calls an oracle to find a weak hypothesis

---

[*]Xinhua Zhang is now at the National ICT Australia (NICTA), Machine Learning Group.

(typically a rank-one matrix) yielding the steepest local reduction of the (unregularized) loss. The associated weight is then determined by accounting for the $\ell_1$ regularization. Our first key contribution is to establish that, when the loss is convex and smooth, the procedure finds an $\epsilon$ accurate solution within $O(1/\epsilon)$ iterations. To the best of our knowledge, this is the first $O(1/\epsilon)$ objective value rate that has been rigorously established for $\ell_1$ *regularized* boosting. [15] considered a similar boosting approach, but required totally corrective updates. In addition, their rate characterizes the diminishment of the gradient, and is $O(1/\epsilon^2)$ as opposed to $O(1/\epsilon)$ established here. [9–11, 16–18] establish similar rates, but only for the constrained version of the problem.

We also show in Section 3 how the empirical performance of $\ell_1$ penalized boosting can be greatly improved by introducing an auxiliary rank-constrained local-optimization within each iteration. Interlacing rank constrained optimization with sparse updates has been shown effective in semi-definite programming [19–21]. [22] applied the idea to trace norm optimization by factoring the reconstruction matrix into two orthonormal matrices and a positive semi-definite matrix. Unfortunately, this strategy creates a very difficult constrained optimization problem, compelling [22] to resort to manifold techniques. Instead, we use a simpler variational representation of matrix norms that leads to a new local objective that is both *unconstrained* and *smooth*. This allows the application of much simpler and much more efficient solvers to greatly accelerate the overall optimization.

Underlying standard sparse approximation methods is an oracle that *efficiently* selects a weak hypothesis (using boosting terminology). Unfortunately these oracle problems are extremely challenging except in limited cases [3, 11]. Our next major contribution, in Section 4, is to formulate an efficient oracle for latent *multiview* factorization models [2, 4], based on a positive semi-definite relaxation that we prove incurs no gap.

Finally, we point out that our focus in this paper is on the optimization of convex problems that relax the "hard" rank constraint. We do *not* explicitly minimize the rank, which is different from [23].

**Notation** We use $\gamma_\mathcal{K}$ to denote the gauge induced by set $\mathcal{K}$; $\|\cdot\|^*$ to denote the dual norm of $\|\cdot\|$; and $\|\cdot\|_F$, $\|\cdot\|_{\mathrm{tr}}$ and $\|\cdot\|_{\mathrm{sp}}$ to denote the Frobenius norm, trace norm and spectral norm respectively. $\|X\|_{R,1}$ denotes the row-wise norm $\sum_i \|X_{i:}\|_R$, while $\langle X, Y \rangle := \mathrm{tr}(X'Y)$ denotes the inner product. The notation $X \succcurlyeq \mathbf{0}$ will denote positive semi-definite; $X_{:i}$ and $X_{i:}$ stands for the $i$-th column and $i$-th row of matrix $X$; and $\mathrm{diag}\{c_i\}$ denotes a diagonal matrix with the $(i,i)$-th entry $c_i$.

## 2  The Boosting Framework with $\ell_1$ Regularization

Consider a coding problem where one is presented an $n \times m$ matrix $Z$, whose columns correspond to $m$ training examples. Our goal is to learn an $n \times k$ dictionary matrix $U$, consisting of $k$ basis vectors, and a $k \times m$ coefficient matrix $V$, such that $UV$ approximates $Z$ under some loss $L(UV)$. We suppress the dependence on the data $Z$ throughout the paper. To remove the scaling invariance between $U$ and $V$, it is customary to restrict the bases, *i.e.* columns of $U$, to the unit ball of some norm $\|\cdot\|_C$. Unfortunately, for a fixed $k$, this coding problem is known to be computationally tractable only for the squared loss. To retain tractability for a variety of convex losses, a popular and successful recent approach has been to avoid any "hard" constraint on the number of bases, *i.e.* $k$, and instead impose regularizers on the matrix $V$ that encourage a low rank or sparse solution.

To be more specific, the following optimization problem lies at the heart of many sparse learning models [*e.g.* 1, 3, 4, 24, 25]:
$$\min_{U: \|U_{:i}\|_C \leq 1} \min_{\tilde{V}} L(U\tilde{V}) + \lambda \|\tilde{V}\|_{R,1}, \tag{1}$$

where $\lambda \geq 0$ specifies the tradeoff between loss and regularization. The $\|\cdot\|_R$ norm in the block $R$-1 norm provides the flexibility of promoting useful structures in the solution, *e.g.* $\ell_1$ norm for sparse solutions, $\ell_2$ norm for low rank solutions, and block structured norms for group sparsity. To solve (1), we first reparameterize the rows of $\tilde{V}$ by $\tilde{V}_{i:} = \sigma_i V_{i:}$, where $\sigma_i \geq 0$ and $\|V_{i:}\|_R \leq 1$. Now (1) can be reformulated by introducing the reconstruction matrix $X := U\tilde{V}$:

$$(1) = \min_X L(X) + \lambda \min_{U, \tilde{V}: \|U_{:i}\|_C \leq 1, U\tilde{V} = X} \|\tilde{V}\|_{R,1} = \min_X L(X) + \lambda \min_{\sigma, U, V: \sigma \geq 0, U\Sigma V = X} \sum_i \sigma_i, \tag{2}$$

where $\Sigma = \mathrm{diag}\{\sigma_i\}$, and $U$ and $V$ in the last minimization also carry norm constraints. (2) is illuminating in two respects. First it reveals that the regularizer essentially seeks a rank-one decomposition of the reconstruction matrix $X$, and penalizes the $\ell_1$ norm of the combination coefficients as a proxy of the "rank". Second, the regularizer in (2) is now expressed precisely in the form of the

| **Algorithm 1:** The vanilla boosting algorithm. | **Algorithm 2:** Boosting with local search. |
|---|---|
| **Require:** The weak hypothesis set $\mathcal{A}$ in (3). | **Require:** A set of weak hypotheses $\mathcal{A}$. |
| 1: Set $X_0 = \mathbf{0}$, $s_0 = 0$. | 1: Set $X_0 = \mathbf{0}$, $U_0 = V_0 = \Lambda_0 = [\ ]$, $s_0 = 0$. |
| 2: **for** $k = 1, 2, \ldots$ **do** | 2: **for** $k = 1, 2, \ldots$ **do** |
| 3: $\quad H_k \leftarrow \underset{H \in \mathcal{A}}{\operatorname{argmin}} \langle \nabla L(X_{k-1}), H \rangle$. | 3: $\quad (\mathbf{u}_k, \mathbf{v}_k) \leftarrow \underset{\mathbf{uv}' \in \mathcal{A}}{\operatorname{argmin}} \langle \nabla L(X_{k-1}), \mathbf{uv}' \rangle$. |
| 4: $\quad (a_k, b_k) \leftarrow$ $\quad\quad \underset{a \geq 0, b \geq 0}{\operatorname{argmin}} L(aX_{k-1} + bH_k) + \lambda(as_k + b)$. | 4: $\quad (a_k, b_k) \leftarrow$ $\quad\quad \underset{a \geq 0, b \geq 0}{\operatorname{argmin}} L(aX_{k-1} + b\,\mathbf{u}_k\mathbf{v}_k') + \lambda(as_k + b)$. |
| 5: $\quad \sigma_i^{(k)} \leftarrow a_k \sigma_i^{(k-1)}, A_i^{(k)} \leftarrow A_i^{(k-1)}, \forall\, i < k$ $\quad\quad \sigma_k^{(k)} \leftarrow b_k, A_k^{(k)} \leftarrow H_k$. | 5: $\quad U_{\text{init}} \leftarrow (\hat{U}_{k-1}\sqrt{a_k \Lambda_{k-1}}, \sqrt{b_k}\mathbf{u}_k)$, $\quad\quad V_{\text{init}} \leftarrow (\sqrt{a_k \Lambda_{k-1}}\hat{V}_{k-1}, \sqrt{b_k}\mathbf{v}_k)'$. |
| 6: $\quad X_k \leftarrow \sum_{i=1}^{k} \sigma_i^{(k)} A_i^{(k)} = a_k X_{k-1} + b_k H_k,$ $\quad\quad s_k \leftarrow \sum_{i=1}^{k} \sigma_i^{(k)} = a_k s_{k-1} + b_k$. | 6: $\quad$ Locally optimize $g(U, V)$ with initial $\quad\quad$ value $(U_{\text{init}}, V_{\text{init}})$. Get a solution $(U_k, V_k)$. |
| 7: **end for** | 7: $\quad X_k \leftarrow U_k V_k, \Lambda_k \leftarrow \operatorname{diag}\{\|U_{:i}\|_C \|V_{i:}\|_R\},$ $\quad\quad s_k \leftarrow \frac{1}{2}\sum_{i=1}^{k}(\|U_{:i}\|_C^2 + \|V_{i:}\|_R^2)$. |
|  | 8: **end for** |

gauge function $\gamma_{\mathcal{K}}$ induced by the convex hull $\mathcal{K}$ of the set[1]

$$\mathcal{A} = \{\mathbf{uv}' : \|\mathbf{u}\|_C \leq 1, \|\mathbf{v}\|_R \leq 1\}. \tag{3}$$

Since $\mathcal{K}$ is convex and symmetric ($-\mathcal{K} = \mathcal{K}$), the gauge function $\gamma_{\mathcal{K}}$ is in fact a norm, hence the support function of $\mathcal{A}$ defines the dual norm $\|\cdot\|$ (see *e.g.* [26, Proposition V.3.2.1]):

$$\|\Lambda\| := \max_{X \in \mathcal{A}} \operatorname{tr}(X'\Lambda) = \max_{\mathbf{u},\mathbf{v}:\|\mathbf{u}\|_C \leq 1, \|\mathbf{v}\|_R \leq 1} \mathbf{u}'\Lambda\mathbf{v} = \max_{\mathbf{u}:\|\mathbf{u}\|_C \leq 1} \|\Lambda'\mathbf{u}\|_R^* = \max_{\mathbf{v}:\|\mathbf{v}\|_R \leq 1} \|\Lambda\mathbf{v}\|_C^*, \tag{4}$$

and the gauge function $\gamma_{\mathcal{K}}$ is simply its dual norm $\|\cdot\|^*$. For example, when $\|\cdot\|_R = \|\cdot\|_C = \|\cdot\|_2$, we have $\|\cdot\| = \|\cdot\|_{\text{sp}}$, so the regularizer (as the dual norm) becomes $\|\cdot\|_{\text{tr}}$. Another special case of this result was found in [4, Theorem 1], where again $\|\cdot\|_R = \|\cdot\|_2$ but $\|\cdot\|_C$ is more complicated than $\|\cdot\|_2$. Note that the original proofs in [1, 4] are somewhat involved. Moreover, this gauge function framework is flexible enough to subsume a number of structurally regularized problems [11, 12], and it is certainly possible to devise other $\|\cdot\|_R$ and $\|\cdot\|_C$ norms that would induce interesting matrix norms.

The gauge function framework also allows us to develop an efficient boosting algorithm for (2), by resorting to the following equivalent problem:

$$\{\sigma_i^*, A_i^*\} := \underset{\sigma_i \geq 0, A_i \in \mathcal{A}}{\operatorname{argmin}} f(\{\sigma_i, A_i\}), \quad \text{where} \quad f(\{\sigma_i, A_i\}) := L\Big(\sum_i \sigma_i A_i\Big) + \lambda \sum_i \sigma_i. \tag{5}$$

The optimal solution $X^*$ of (2) can be easily recovered as $\sum_i \sigma_i^* A_i^*$. Note that in the boosting terminology, $\mathcal{A}$ corresponds to the set of weak hypotheses.

## 2.1 The boosting algorithm

To solve (5) we propose the boosting strategy presented in Algorithm 1. At each iteration, a weak hypothesis $H_k$ that yields the most rapid local decrease of the loss $L$ is selected. Then $H_k$ is combined with the previous ensemble by tuning its weights to optimize the regularized objective. Note that in Step 5 all the weak hypotheses selected in the previous steps are scaled by the *same* value.

As the $\ell_1$ regularizer requires the sum of all the weights, we introduce a variable $s_k$ that recursively updates this sum in Step 6. In addition, $X_k$ is used only in Step 3 and 4, which do not require its explicit expansion in terms of the elements of $\mathcal{A}$. Therefore this expansion of $X_k$ does not need to be explicitly maintained and Step 5 is included only for conceptual clarity.

## 2.2 Rate of convergence

We prove the convergence rate of Algorithm 1, under the standard assumption:

**Assumption 1** *$L$ is bounded from below and has bounded sub-level sets. The problem (5) admits at least one minimizer $X^*$. $L$ is differentiable and satisfies the following inequality for all $\eta \in$*

$[0, 1]$ *and $A, B$ in the (smallest) convex set that contains both $X^*$ and the sub-level set of $f(\mathbf{0})$:*
$L((1 - \eta)A + \eta B) \leq L(A) + \eta \langle B - A, \nabla L(A) \rangle + \frac{C_L \eta^2}{2}$. *Here $C_L > 0$ is a finite constant that depends only on $L$ and $X^*$.*

**Theorem 1 (Rate of convergence)** *Under Assumption 1, Algorithm 1 finds an $\epsilon$ accurate solution to (5) in $O(1/\epsilon)$ steps. More precisely, denoting $f^*$ as the minimum of (5), then*

$$f(\{\sigma_i^{(k)}, A_i^{(k)}\}) - f^* \leq \frac{4C_L}{k + 2}. \tag{6}$$

The proof is given in Appendix A. Note that the rate is independent of the regularization constant $\lambda$.

In the proof we fix the variable $a$ in Step 4 of Algorithm 1 to be simply $\frac{2}{k+2}$; it should be clear that setting $a$ by line search will only accelerate the convergence. An even more aggressive scheme is the totally corrective update [15], which in Step 4 finds the weights for *all* $A_i^{(k)}$'s selected so far:

$$\min_{\sigma_i \geq 0} L\left(\sum_{i=1}^{k} \sigma_i A_i^{(k)}\right) + \lambda \sum_{i=1}^{k} \sigma_i. \tag{7}$$

But in this case we will have to explicitly maintain the expansion of $X_t$ in terms of the $A_i^{(k)}$'s. For boosting without regularization, the $1/\epsilon$ rate of convergence is known to be optimal [27]. We conjecture that $1/\epsilon$ is also a lower bound for regularized boosting.

**Extensions** Our proof technique allows the regularizer to be generalized to the form $h(\gamma_{\mathcal{K}}(X))$, where $h$ is a convex non-decreasing function over $[0, \infty)$. In (5), this replaces $\sum_i \sigma_i$ with $h(\sum_i \sigma_i)$. By taking $h(x)$ as the indicator $h(x) = 0$ if $x \leq 1; \infty$ otherwise, our rate can be straightforwardly translated into the constrained setting.

## 3 Local Optimization with Fixed Rank

In Algorithm 1, $X_k$ is determined by searching in the conic hull of $X_{k-1}$ and $H_k$.[2] Suppose there exists some auxiliary procedure that allows $X_k$ to be further improved somehow to $Y_k$ (*e.g.* by local greedy search), then the overall optimization can benefit from it. The only challenge, nevertheless, is how to restore the "context" from $Y_k$, especially the bases $A_i$ and their weights $\sigma_i$.

In particular, suppose we have an auxiliary function $g$ and the following procedure is feasible:

**1.** Initialization: given an ensemble $\{\sigma_i, A_i\}$, there exists a $S$ such that $g(S) \leq f(\{\sigma_i, A_i\})$.

**2.** Local optimization: some (local) optimizer can find a $T$ such that $g(T) \leq g(S)$.

**3.** Recovery: one can recover an ensemble $\{\beta_i, B_i : \beta_i \geq 0, B_i \in \mathcal{A}\}$ such that $f(\{\beta_i, B_i\}) \leq g(T)$.

Then obviously the new ensemble $\{\beta_i, B_i\}$ improves upon $\{\sigma_i, A_i\}$. This local search scheme can be easily embedded into Algorithm 1 as follows. After Step 5, initialize $S$ by $\{\sigma_i^{(k)}, A_i^{(k)}\}$. Perform local optimization and recover $\{\beta_i, B_i\}$. Then replace Step 6 by $X_k = \sum_i \beta_i B_i$ and $s_k = \sum_i \beta_i$. The rate of convergence will directly carry over. However, the major challenge here is the potentially expensive step of recovery because little assumption or constraint is made on $T$.

Fortunately, a careful examination of Algorithm 1 reveals that a complete recovery of $\{\beta_i, B_i\}$ is not required. Indeed, only two "sufficient statistics" are needed: $X_k$ and $s_k$, and therefore it suffices to recover them only. Next we will show how this can be accomplished efficiently in (2) . Two simple propositions will play a key role. Both proofs can be found in Appendix C.

**Proposition 1** *For the gauge $\gamma_{\mathcal{K}}$ induced by $\mathcal{K}$, the convex hull of $\mathcal{A}$ in (3), we have*

$$\gamma_{\mathcal{K}}(X) = \min_{U,V : UV = X} \frac{1}{2} \sum_i \left( \|U_{:i}\|_C^2 + \|V_{i:}\|_R^2 \right). \tag{8}$$

If $\|\cdot\|_R = \|\cdot\|_C = \|\cdot\|_2$, then $\gamma_{\mathcal{K}}$ becomes the trace norm (as we saw before), and $\sum_i(\|U_{:i}\|_C^2 + \|V_{i:}\|_R^2)$ is simply $\|U\|_F^2 + \|V\|_F^2$. Then Proposition 1 is a well-known variational form of the trace norm [28]. This motivates us to choose the auxiliary function as

$$g(U, V) = L(UV) + \frac{\lambda}{2} \sum_i \left( \|U_{:i}\|_C^2 + \|V_{i:}\|_R^2 \right). \tag{9}$$

**Proposition 2** *For any $U \in \mathbb{R}^{m \times k}$ and $V \in \mathbb{R}^{k \times n}$, there exist $\sigma_i \geq 0$, $\mathbf{u}_i \in \mathbb{R}^m$, and $\mathbf{v}_i \in \mathbb{R}^n$ such that*

$$UV = \sum_{i=1}^k \sigma_i \mathbf{u}_i \mathbf{v}_i', \quad \|\mathbf{u}_i\|_C \leq 1, \quad \|\mathbf{v}_i\|_R \leq 1, \quad \sum_{i=1}^k \sigma_i = \frac{1}{2} \sum_{i=1}^k \left( \|U_{:i}\|_C^2 + \|V_{i:}\|_R^2 \right). \tag{10}$$

Now we can specify concrete details for local optimization in the context of matrix norms:

**1.** Initialize: given $\{\sigma_i \geq 0, \mathbf{u}_i \mathbf{v}_i' \in \mathcal{A}\}_{i=1}^k$, set $(U_{\text{init}}, V_{\text{init}})$ to satisfy $g(U_{\text{init}}, V_{\text{init}}) = f(\{\sigma_i, \mathbf{u}_i \mathbf{v}_i'\})$:

$$U_{\text{init}} = (\sqrt{\sigma_1} \mathbf{u}_1, \dots, \sqrt{\sigma_k} \mathbf{u}_k), \qquad \text{and} \qquad V_{\text{init}} = (\sqrt{\sigma_1} \mathbf{v}_1, \dots, \sqrt{\sigma_k} \mathbf{v}_k)'. \tag{11}$$

**2.** Locally optimize $g(U, V)$ with initialization $(U_{\text{init}}, V_{\text{init}})$, to obtain a solution $(U^*, V^*)$.

**3.** Recovery: use Proposition 2 to (conceptually) recover $\{\beta_i, \hat{\mathbf{u}}_i, \hat{\mathbf{v}}_i\}$ from $(U^*, V^*)$.

The key advantage of this procedure is that Proposition 2 allows $X_k$ and $s_k$ to be computed *directly* from $(U^*, V^*)$, keeping the recovery completely implicit:

$$X_k = \sum_{i=1}^k \beta_i \hat{\mathbf{u}}_i \hat{\mathbf{v}}_i' = U^* V^*, \quad \text{and} \quad s_k = \sum_{i=1}^k \sigma_i = \frac{1}{2} \sum_{i=1}^k \left( \|U_{:i}^*\|_C^2 + \|V_{i:}^*\|_R^2 \right). \tag{12}$$

In addition, Proposition 2 ensures that locally improving the solution does not incur an increment in the number of weak hypotheses. Using the same trick, the $(U_{\text{init}}, V_{\text{init}})$ in (11) for the $(k+1)$-th iteration can also be formulated in terms of $(U^*, V^*)$. Different from the local optimization for trace norm in [21] which naturally works on the original objective, our scheme requires a nontrivial (variational) reformulation of the objective based on Propositions 1 and 2.

The final algorithm is summarized in Algorithm 2, where $\hat{U}$ and $\hat{V}$ in Step 5 denote the column-wise and row-wise normalized versions of $U$ and $V$, respectively. Compared to the local optimization in [22], which is hampered by orthogonal and PSD constraints, our (local) objective in (9) is unconstrained and smooth for many instances of $\|\cdot\|_C$ and $\|\cdot\|_R$. This is plausible because no other constraints (besides the norm constraint), such as orthogonality, are imposed on $U$ and $V$ in Proposition 2. Thus the local optimization we face, albeit non-convex in general, is more amenable to efficient solvers such as L-BFGS.

**Remark** Consider if one performs totally corrective update as in (7). Then all of the coefficients and weak hypotheses from $(U^*, V^*)$ have to be considered, which can be computationally expensive. For example, in the case of trace norm, this leads to a full SVD on $U^* V^*$. Although $U^*$ and $V^*$ usually have low rank, which can be exploited to ameliorate the complexity, it is clearly preferable to completely eliminate the recovery step, as in Algorithm 2.

## 4 Latent Generative Model with Multiple Views

Underlying most boosting algorithms is an oracle that identifies the steepest descent weak hypothesis (Step 3 of Algorithm 1). Approximate solutions often suffice [8, 9]. When $\|\cdot\|_R$ and $\|\cdot\|_C$ are both Euclidean norms, this oracle can be efficiently computed via the leading left and right singular vector pair. However, for most other interesting cases like low rank tensors, such an oracle is intractable [29]. In this section we discover that for an important problem of multiview learning, the oracle can be surprisingly solved in polynomial time, yielding an efficient computational strategy.

Multiview learning analyzes multi-modal data, such as heterogeneous descriptions of text, image and video, by exploiting the implicit conditional independence structure. In this case, beyond a single dictionary $U$ and coefficient matrix $V$ that model a single view $Z^{(1)}$, multiple dictionaries $U^{(k)}$ are needed to reconstruct multiple views $Z^{(k)}$, while keeping the latent representation $V$ shared across all views. Formally the problem in multiview factorization is to optimize [2, 4]:

$$\min_{U^{(1)}: \|U_{:i}^{(1)}\|_C \leq 1} \cdots \min_{U^{(k)}: \|U_{:i}^{(k)}\|_C \leq 1} \min_V \sum_{t=1}^k L_t(U^{(t)} V) + \lambda \|V\|_{R,1}. \tag{13}$$

We can easily re-express the problem as an equivalent "single" view formulation (1) by stacking all $\{U^{(t)}\}$ into the rows of a big matrix $U$, with a new column norm $\|U_{:i}\|_C := \max_{t=1...k} \|U^{(t)}_{:i}\|_C$. Then the constraints on $U^{(t)}$ in (13) can be equivalently written as $\|U_{:i}\|_C \leq 1$, and Algorithm 2 can be directly applied with two specializations. First the auxiliary function $g(U,V)$ in (9) becomes

$$g(U,V) = L(UV) + \frac{\lambda}{2}\sum_i \left(\left(\max_{t=1...k}\|U^{(t)}_{:i}\|_C\right)^2 + \|V_{i:}\|_R^2\right) = L(UV) + \frac{\lambda}{2}\sum_i\left(\max_{t=1...k}\|U^{(t)}_{:i}\|_C^2 + \|V_{i:}\|_R^2\right)$$

which can be locally optimized. The only challenge left is the oracle problem in (4), which takes the following form when all norms are Euclidean:

$$\max_{\|\mathbf{u}\|_C \leq 1, \|\mathbf{v}\| \leq 1} \mathbf{u}'\Lambda\mathbf{v} = \max_{\|\mathbf{u}\|_C \leq 1}\|\Lambda'\mathbf{u}\|^2 = \max_{\mathbf{u}:\forall t, \|\mathbf{u}_t\| \leq 1}\left\|\sum_t \Lambda'_t\mathbf{u}_t\right\|^2. \tag{14}$$

[4, 24] considered the case where $k = 2$ and showed that exact solutions to (14) can be found efficiently. But their derivation does not seem to extend to $k > 2$. Fortunately there is still an interesting and tractable scenario. Consider multilabel classification with a small number of classes, and $U^{(1)}$ and $U^{(2)}$ are two views of features (*e.g.* image and text). Then each class label corresponds to a view and the corresponding $u_t$ is univariate. Since there must be an optimal solution on the extreme points of the feasible region, we can enumerate $\{-1, 1\}$ for $u_t$ ($t \geq 3$) and for each assignment solve a subproblem of the following form that instantiates (14) ($\mathbf{c}$ is a constant vector)

$$(QP) \quad \max_{\mathbf{u}_1, \mathbf{u}_2} \|\Lambda'_1\mathbf{u}_1 + \Lambda'_2\mathbf{u}_2 + \mathbf{c}\|^2, \ s.t. \ \|\mathbf{u}_1\| \leq 1, \|\mathbf{u}_2\| \leq 1. \tag{15}$$

Due to inhomogeneity, the technique in [4] is not applicable. Rewrite (15) in matrix form

$$(QP) \quad \min_{\mathbf{z}} \langle M_0, \mathbf{z}\mathbf{z}'\rangle \quad s.t. \quad \langle M_1, \mathbf{z}\mathbf{z}'\rangle \leq 0 \quad \langle M_2, \mathbf{z}\mathbf{z}'\rangle \leq 0 \quad \langle I_{00}, \mathbf{z}\mathbf{z}'\rangle = 1, \tag{16}$$

where $\mathbf{z} = \begin{pmatrix} r \\ \mathbf{u}_1 \\ \mathbf{u}_2 \end{pmatrix}$, $M_0 = -\begin{pmatrix} 0 & \mathbf{c}'\Lambda'_1 & \mathbf{c}'\Lambda'_2 \\ \Lambda_1\mathbf{c} & \Lambda_1\Lambda'_1 & \Lambda_1\Lambda'_2 \\ \Lambda_2\mathbf{c} & \Lambda_2\Lambda'_1 & \Lambda_2\Lambda'_2 \end{pmatrix}$, $M_1 = \begin{pmatrix} -1 & & \\ & I & \\ & & \mathbf{0} \end{pmatrix}$, $M_2 = \begin{pmatrix} -1 & & \\ & \mathbf{0} & \\ & & I \end{pmatrix}$,

and $I_{00}$ is a zero matrix with only the $(1, 1)$-th entry being 1. Let $X = \mathbf{z}\mathbf{z}'$, a semi-definite programming relaxation for $(QP)$ can be obtained by dropping the rank-one constraint:

$$(SP) \quad \min_{X} \langle M_0, X\rangle, \quad s.t. \quad \langle M_1, X\rangle \leq 0, \quad \langle M_2, X\rangle \leq 0, \quad \langle I_{00}, X\rangle = 1, \quad X \succeq \mathbf{0}. \tag{17}$$

Its dual problem, which is also the Lagrange dual of $(QP)$, can be written as

$$(SD) \quad \max_{y_0, y_1, y_2} y_0, \quad s.t. \quad Z := M_0 - y_0 I_{00} + y_1 M_1 + y_2 M_2 \succeq \mathbf{0}, \quad y_1 \geq 0, \quad y_2 \geq 0. \tag{18}$$

$(SD)$ is a convex problem that can be solved efficiently by, *e.g.*, cutting plane methods. $(SP)$ is also a convex semidefinite program (SDP) amenable for standard SDP solvers. However further recovering the solution to $(QP)$ is *not* straightforward, because there may be a gap between the optimal values of $(SP)$ and $(QP)$. The gap is zero (*i.e.* strong duality between $(QP)$ and $(SD)$) only if the rank-one constraint that $(SP)$ dropped from $(QP)$ is automatically satisfied, *i.e.* if $(SP)$ has a rank-one optimal solution.

Fortunately, as one of our main results, we prove that strong duality always holds for the particular problem originating from (15). Our proof utilizes some recent development in optimization [30], and is relegated to Appendix D.

## 5  Experimental Results

We compared our Algorithm 2 with three state-of-the-art solvers for trace norm regularized objectives: MMBS[3] [22], DHM [15], and JS [8]. JS was proposed for solving the constrained problem: $\min_X L(X)$ s.t. $\|X\|_{\text{tr}} \leq \zeta$, which makes it hard to compare with solvers for the penalized problem: $\min_X L(X) + \lambda\|X\|_{\text{tr}}$. As a workaround, we first chose a $\lambda$, and found the optimal solution $X^*$ for the penalized problem. Then we set $\zeta = \|X^*\|_{\text{tr}}$ and finally solved the constrained problem by JS. In this case, it is only fair to compare how fast $L(X)$ (loss) is decreased by various solvers, rather than $L(X) + \lambda\|X\|_*$ (objective). DHM is sensitive to the estimate of the Lipschitz constant of the gradient of $L$, which we manually tuned for a small value such that DHM still converges. Since the code for MMBS is specialized to matrix completion, it was used only in this comparison. Traditional solvers such as proximal methods [6] were not included because they are much slower.

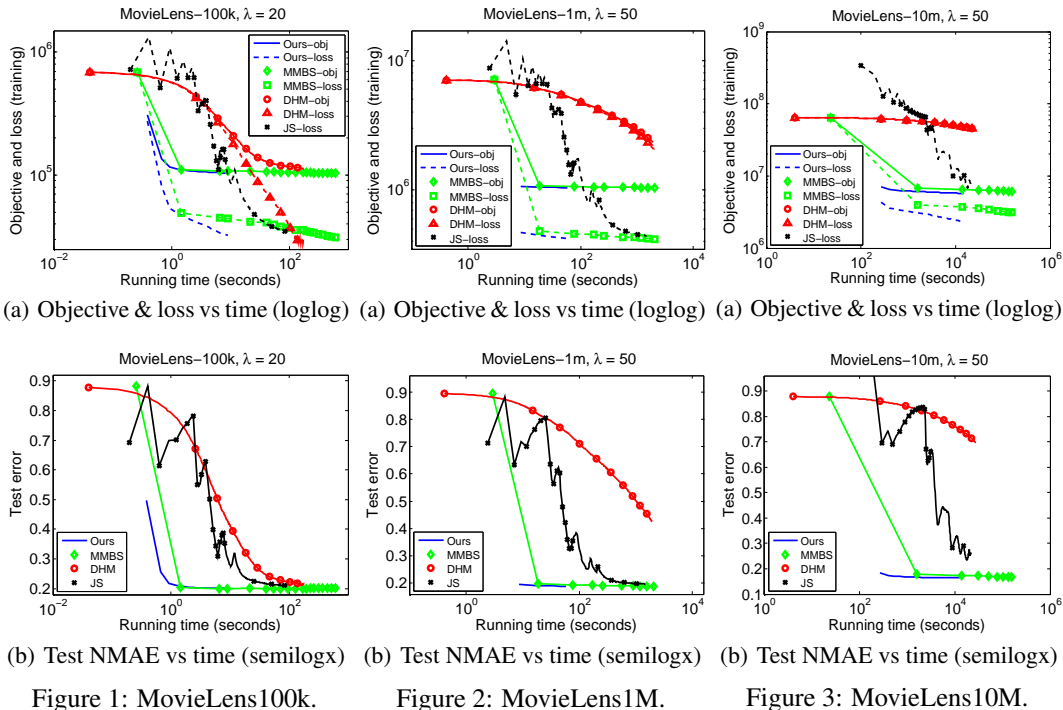

(a) Objective & loss vs time (loglog)  (a) Objective & loss vs time (loglog)  (a) Objective & loss vs time (loglog)

(b) Test NMAE vs time (semilogx)  (b) Test NMAE vs time (semilogx)  (b) Test NMAE vs time (semilogx)

Figure 1: MovieLens100k.  Figure 2: MovieLens1M.  Figure 3: MovieLens10M.

**Comparison 1: Matrix completion**   We first compared all methods on a matrix completion problem, using the standard datasets MovieLens100k, MovieLens1M, and MovieLens10M [6, 8, 21], which are sized $943 \times 1682$, $6040 \times 3706$, and $69878 \times 10677$ respectively (#user $\times$ #movie). They contain $10^5$, $10^6$ and $10^7$ movie ratings valued from 1 to 5, and the task is to predict the rating for a user on a movie. The training set was constructed by randomly selecting 50% ratings for each user, and the prediction is made on the rest 50% ratings. In Figure 1 to 3, we show how fast various algorithms drive down the training objective, training loss $L$ (squared Euclidean distance), and the normalized mean absolute error (NMAE) on the test data [see, *e.g.*, 6, 8]. We tuned the $\lambda$ to optimize the test NMAE.

From Figure 1(a), 2(a), 3(a), it is clear that it takes much less amount of CPU time for our method to reduce the objective value (solid line) and the loss $L$ (dashed line). This implies that local search and partially corrective updates in our method are very effective. Not surprisingly MMBS is the closest to ours in terms of performance because it also adopts local optimization. However it is still slower because their local search is conducted on a *constrained* manifold. In contrast, our local search objective is entirely unconstrained and smooth, which we manage to solve efficiently by L-BFGS.[4]

JS, though applied indirectly, is faster than DHM in reducing the loss. We observed that DHM kept running coordinate descent with a constant step size, while the totally corrective update was rarely taken. We tried accelerating it by using a smaller value of the estimate of the Lipschitz constant of the gradient of $L$, but it leads to divergence after a rapid decrease of the objective for the first few iterations. A hybrid approach might be useful.

We also studied the evolution of the NMAE performance on the test data. For this we compared the matrix reconstruction after each iteration against the ground truth. As plotted in Figure 1(b), 2(b), 3(b), our approach achieves comparable (or better) NMAE in much less time than all other methods.

**Comparison 2: multitask and multiclass learning**   Secondly, we tested on a multiclass classification problem with synthetic dataset. Following [15], we generated a dataset of $D = 250$ features and $C = 100$ classes. Each class $c$ has 10 training examples and 10 test examples drawn independently and identically from a class-specific multivariate Gaussian $\mathcal{N}(\boldsymbol{\mu}_c, \Sigma_c)$. $\boldsymbol{\mu}_c \in \mathbb{R}^{250}$ has the last 200 coordinates being 0, and the top 50 coordinates were chosen uniformly random from $\{-1, 1\}$. The $(i, j)$-th element of $\Sigma_c$ is $2^2(0.5)^{|i-j|}$. The task is to predict the class membership of a given example. We used the logistic loss for a model matrix $W \in \mathbb{R}^{D \times C}$. In particular, for each

training example $\mathbf{x}_i$ with label $y_i \in \{1, .., C\}$, we defined an individual loss $L_i(W)$ as

$$L_i(W) = -\log p(y_i|\mathbf{x}_i; W),$$

where for any class $c$,

$$p(c|\mathbf{x}_i; W) = Z_i^{-1}\exp(W'_{:c}\mathbf{x}_i),$$
$$Z_i = \sum_c \exp(W'_{:c}\mathbf{x}_i).$$

Then $L(W)$ is defined as the average of $L_i(W)$ over the whole training set. We found that $\lambda = 0.01$ yielded the lowest test classification error; the corresponding results are given in Figure 4. Clearly, the intermediate models output by our scheme achieve comparable (or better) training objective and test error in orders of magnitude less time than those generated by DHM and JS.

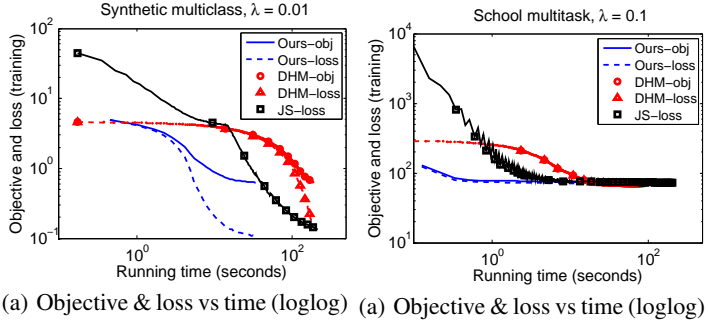

(a) Objective & loss vs time (loglog)   (a) Objective & loss vs time (loglog)

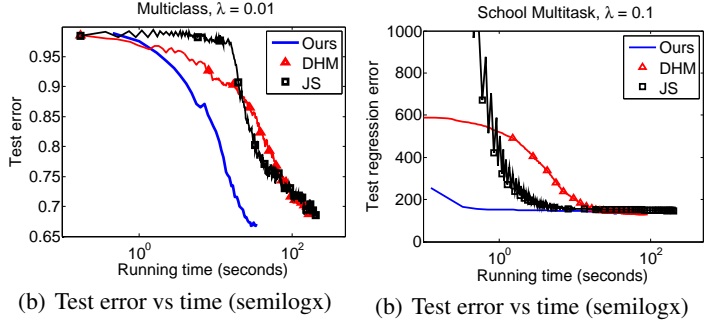

(b) Test error vs time (semilogx)   (b) Test error vs time (semilogx)

Figure 4: Multiclass classification with synthetic datset.

Figure 5: Multitask learning for school dataset.

We also applied the solvers to a multitask learning problem with the school dataset [25]. The task is to predict the score of 15362 students from 139 secondary schools based on a number of school-specific and student-specific attributes. Each school is considered as a task for which a predictor is learned. We used the first random split of training and testing data provided by [25] [5], and set $\lambda$ so as to achieve the lowest test squared error. Again, as shown in Figure 5 our approach is much faster than DHM and JS in finding the optimal solution for training objective and test error. As the problem requires a large $\lambda$, the trace norm penalty is small, making the loss close to the objective.

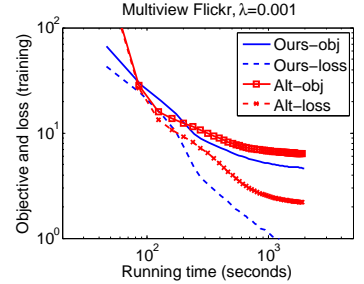

Figure 6: Multiview training.

**Comparison 3: Multiview learning**   Finally we perform an initial test on our global optimization technique for learning latent models with multiple views. We used the Flickr dataset from NUS-WIDE [31]. Its first view is a 634 dimensional low-level feature, and the second view consists of 1000 dimensional tags. The class labels correspond to the type of animals and we randomly chose 5 types with 20 examples in each type. The task is to train the model in (13) with $\lambda = 10^{-3}$. We used squared loss for the first view, and logistic loss for the other views.

We compared our method with a local optimization approach to solving (13). The local method first fixes all $U^{(t)}$ and minimizes $V$, which is a convex problem that can be solved by FISTA [32]. Then it fixes $V$ and optimizes $U^{(t)}$, which is again convex. We let Alt refer to the scheme that alternates these updates to convergence. From Figure 6 it is clear that Alt is trapped by a locally optimal solution, which is inferior to a globally optimal solution that our method is guaranteed to find. Our method also reduces both the objective and the loss slightly faster than Alt.

## 6   Conclusion and Outlook

We have proposed a new boosting algorithm for a wide range of matrix norm regularized problems. It is closely related to generalized conditional gradient method [33]. We established the $O(1/\epsilon)$ convergence rate, and showed its empirical advantage over state-of-the-art solvers on large scale problems. We also applied the method to a novel problem, latent multiview learning, for which we designed a new efficient oracle. We plan to study randomized boosting with $\ell_1$ regularization [34] , and to extend the framework to more general nonlinear regularization [3].

## Footnotes

[1]Recall that the gauge function $\gamma_{\mathcal{K}}(X)$ is defined as $\gamma_{\mathcal{K}}(X) := \inf\{\sum_i \sigma_i : \sum_i \sigma_i A_i = X,\ A_i \in \mathcal{K},\ \sigma_i \geq 0\}$.

[2] This does *not* mean $X_k$ is a minimizer of $L(X) + \lambda\gamma_{\mathcal{K}}(X)$ in that cone, because the bases are not optimized simultaneously. Incidentally, this also shows why working with (5) turns out to be more convenient.

[3] http://www.montefiore.ulg.ac.be/~mishra/softwares/traceNorm.html

[4] http://www.cs.ubc.ca/~pcarbo/lbfgsb-for-matlab.html

[5] http://ttic.uchicago.edu/~argyriou/code/mtl_feat/school_splits.tar

# References

[1] F. Bach, J. Mairal, and J. Ponce. Convex sparse matrix factorizations. arXiv:0812.1869v1, 2008.

[2] H. Lee, R. Raina, A. Teichman, and A. Ng. Exponential family sparse coding with application to self-taught learning. In *IJCAI*, 2009.

[3] D. Bradley and J. Bagnell. Convex coding. In *UAI*, 2009.

[4] X. Zhang, Y-L Yu, M. White, R. Huang, and D. Schuurmans. Convex sparse coding, subspace learning, and semi-supervised extensions. In *AAAI*, 2011.

[5] T. K. Pong, P. Tseng, S. Ji, and J. Ye. Trace norm regularization: Reformulations, algorithms, and multi-task learning. *SIAM Journal on Optimization*, 20(6):3465–3489, 2010.

[6] K-C Toh and S. Yun. An accelerated proximal gradient algorithm for nuclear norm regularized least squares problems. *Pacific Journal of Optimization*, 6:615–640, 2010.

[7] J-F Cai, E. J. Candés, and Z. Shen. A singular value thresholding algorithm for matrix completion. *SIAM Journal on Optimization*, 20(4):1956–1982, 2010.

[8] M. Jaggi and M. Sulovsky. A simple algorithm for nuclear norm regularized problems. In *ICML*, 2010.

[9] E. Hazan. Sparse approximate solutions to semidefinite programs. In *LATIN*, 2008.

[10] K. L. Clarkson. Coresets, sparse greedy approximation, and the Frank-Wolfe algorithm. In *SODA*, 2008.

[11] A. Tewari, P. Ravikumar, and I. S. Dhillon. Greedy algorithms for structurally constrained high dimensional problems. In *NIPS*, 2011.

[12] V. Chandrasekaran, B. Recht, P. A. Parrilo, and A. S. Willsky. The convex geometry of linear inverse problems. *Foundations of Computational Mathematics*, 12(6):805–849, 2012.

[13] Y. Bengio, N.L. Roux, P. Vincent, O. Delalleau, and P. Marcotte. Convex neural networks. In *NIPS*, 2005.

[14] L. Mason, J. Baxter, P. L. Bartlett, and M. Frean. Functional gradient techniques for combining hypotheses. In *Advances in Large Margin Classifiers*, pages 221–246, Cambridge, MA, 2000. MIT Press.

[15] M. Dudik, Z. Harchaoui, and J. Malick. Lifted coordinate descent for learning with trace-norm regularizations. In *AISTATS*, 2012.

[16] S. Shalev-Shwartz, N. Srebro, and T. Zhang. Trading accuracy for sparsity in optimization problems with sparsity constraints. *SIAM Journal on Optimization*, 20:2807–2832, 2010.

[17] X. Yuan and S. Yan. Forward basis selection for sparse approximation over dictionary. In *AISTATS*, 2012.

[18] T. Zhang. Sequential greedy approximation for certain convex optimization problems. *IEEE Trans. Information Theory*, 49(3):682–691, 2003.

[19] S. Burer and R. Monteiro. Local minima and convergence in low-rank semidefinite programming. *Mathematical Programming*, 103(3):427–444, 2005.

[20] M. Journee, F. Bach, P.-A. Absil, and R. Sepulchre. Low-rank optimization on the cone of positive semidefinite matrices. *SIAM Journal on Optimization*, 20:2327C–2351, 2010.

[21] S. Laue. A hybrid algorithm for convex semidefinite optimization. In *ICML*, 2012.

[22] B. Mishra, G. Meyer, F. Bach, and R. Sepulchre. Low-rank optimization with trace norm penalty. Technical report, 2011. http://arxiv.org/abs/1112.2318.

[23] S. Shalev-Shwartz, A. Gonen, and O. Shamir. Large-scale convex minimization with a low-rank constraint. In *ICML*, 2011.

[24] M. White, Y. Yu, X. Zhang, and D. Schuurmans. Convex multi-view subspace learning. In *NIPS*, 2012.

[25] A. Argyriou, T. Evgeniou, and M. Pontil. Convex multi-task feature learning. *Machine Learning*, 73(3): 243–272, 2008.

[26] J-B Hiriart-Urruty and C. Lemaréchal. *Convex Analysis and Minimization Algorithms, I and II*, volume 305 and 306. Springer-Verlag, 1993.

[27] I. Mukherjee, C. Rudin, and R. Schapire. The rate of convergence of Adaboost. In *COLT*, 2011.

[28] N. Srebro, J. Rennie, and T. Jaakkola. Maximum-margin matrix factorization. In *NIPS*, 2005.

[29] C. Hillar and L-H Lim. Most tensor problems are NP-hard. arXiv:0911.1393v3, 2012.

[30] W. Ai and S. Zhang. Strong duality for the CDT subproblem: A necessary and sufficient condition. *SIAM Journal on Optimization*, 19:1735–1756, 2009.

[31] T.S. Chua, J. Tang, R. Hong, H. Li, Z. Luo, and Y.T. Zhang. A real-world web image database from national university of singapore. In *International Conference on Image and Video Retrieval*, 2009.

[32] A. Beck and M. Teboulle. A fast iterative shrinkage-thresholding algorithm for linear inverse problems. *SIAM Journal on Imaging Sciences*, 2(1):183–202, 2009.

[33] K. Bredies, D. Lorenz, and P. Maass. A generalized conditional gradient method and its connection to an iterative shrinkage method. *Computational Optimization and Applications*, 42:173–193, 2009.

[34] Y. Nesterov. Efficiency of coordinate descent methods on huge-scale optimization problems. *SIAM Journal on Optimization*, 22(2):341–362, 2012.

